# The Parti-game Algorithm for Variable Resolution Reinforcement Learning in Multidimensional State-spaces

**Andrew W. Moore**
School of Computer Science
Carnegie-Mellon University
Pittsburgh, PA 15213

## Abstract

Parti-game is a new algorithm for learning from delayed rewards in high dimensional real-valued state-spaces. In high dimensions it is essential that learning does not explore or plan over state space uniformly. Parti-game maintains a decision-tree partitioning of state-space and applies game-theory and computational geometry techniques to efficiently and reactively concentrate high resolution only on critical areas. Many simulated problems have been tested, ranging from 2-dimensional to 9-dimensional state-spaces, including mazes, path planning, non-linear dynamics, and uncurling snake robots in restricted spaces. In all cases, a good solution is found in less than twenty trials and a few minutes.

## 1 REINFORCEMENT LEARNING

Reinforcement learning [Samuel, 1959, Sutton, 1984, Watkins, 1989, Barto *et al.*, 1991] is a promising method for control systems to program and improve themselves. This paper addresses its biggest stumbling block: the curse of dimensionality [Bellman, 1957], in which costs increase exponentially with the number of state variables.

Some earlier work [Simons *et al.*, 1982, Moore, 1991, Chapman and Kaelbling, 1991, Dayan and Hinton, 1993] has considered recursively partitioning state-space while learning from delayed rewards. The new ideas in the parti-game algorithm in-

clude (i) a game-theoretic splitting criterion to robustly choose spatial resolution (ii) real-time incremental maintenance and planning with a database of all previous experiences, and (iii) using local greedy controllers for high-level "funneling" actions.

## 2   ASSUMPTIONS

The parti-game algorithm applies to difficult learning control problems in which:

1. State and action spaces are continuous and multidimensional.
2. "Greedy" and hill-climbing techniques would become stuck, never attaining the goal.
3. Random exploration would be hopelessly time-consuming.
4. The system dynamics and control laws can have discontinuities and are unknown: they must be learned.

The experiments reported later all have properties 1–4. However, the initial algorithm, described and tested here, has the following restrictions:

5. Dynamics are deterministic.
6. The task is specified by a goal, not an arbitrary reward function.
7. The goal state is known.
8. A "good" solution is required, not necessarily the optimal path. This notion of goodness can be formalized as "the optimal path to within a given resolution of state space".
9. A local greedy controller is available, which we can ask to move greedily towards any desired state. There is no guarantee that a request to the greedy controller will succeed. For example, in a maze a greedy path to the goal would quickly hit a wall.

Future developments may include relatively straightforward additions to the algorithm that would remove the need for restrictions 6–9. Restriction 5 is harder to remove.

## 3   ESSENTIALS OF THE PARTI-GAME ALGORITHM

The state space is broken into partitions by a kd-tree [Friedman *et al.*, 1977]. The controller can always sense its current (continuous valued) state, and can cheaply compute which partition it is in. The space of actions is also discretized so that in a partition with $N$ neighboring partitions, there are $N$ high-level actions. Each high level action corresponds to a local greedy controller, aiming for the center of the corresponding neighboring partition.

Each partition keeps records of all the occasions on which the system state has passed through it. Along with each record is a memory of which high level action was used (i.e. which neighbor was aimed for) and what the outcome was. Figure 1 provides an illustration.

Given this database of (partition, high-level-action, outcome) triplets, and our knowledge of the partition containing the goal state, we can try to compute the

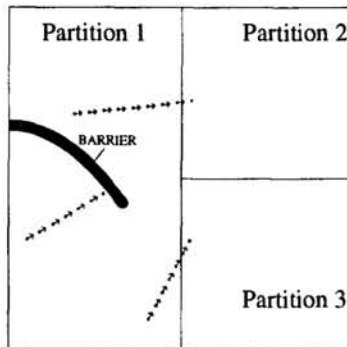

Figure 1: Three trajectories starting in partition 1, using high-level action "Aim at partition 2". Partition 1 remembers three outcomes.
(Part 1, Aim 2 $\rightarrow$ Part 2)
(Part 1, Aim 2 $\rightarrow$ Part 1)
(Part 1, Aim 2 $\rightarrow$ Part 3)

best route to the goal. The standard approach would be to model the system as a Markov Decision Task in which we empirically estimate the partition transition probabilities. However, the probabilistic interpretation of coarse resolution partitions can lead to policies which get stuck. Instead, we use a game-theoretic approach, in which we imagine an adversary. This adversary sees our choice of high-level action, and is allowed to select any of the observed previous outcomes of the action in this partition. Partitions are scored by minimaxing: the adversary plays to delay or prevent us getting to the goal and we play to get to the goal as quickly as possible.

Whenever the system's continuous state passes between partitions, the database of state transitions is updated and, if necessary, the minimax scores of all partitions are updated. If real-time constraints do not permit full recomputation, the updates take place incrementally in a manner similar to prioritized sweeping [Moore and Atkeson, 1993].

As well as being robust to coarseness, the game-theoretic approach also tells us where we should increase the resolution. Whenever we compute that we are in a losing partition we perform resolution increase. We first compute the complete set of connected partitions which are also losing partitions. We then find the subset of these partitions which border some non-losing region. We increase the resolution of all these border states by splitting them along their longest axes[1].

## 4   INITIAL EXPERIMENTS

Figure 2 shows a 2-d continuous maze. Figure 3 shows the performance of the robot during the very first trial. It begins with intense exploration to find a route out of the almost entirely enclosed start region. Having eventually reached a sufficiently high resolution, it discovers the gap and proceeds greedily towards the goal, only to be stopped by the goal's barrier region. The next barrier is traversed at a much lower resolution, mainly because the gap is larger.

Figure 4 shows the second trial, started from a slightly different position. The policy derived from the first trial gets us to the goal without further exploration. The trajectory has unnecessary bends. This is because the controller is discretized according to the current partitioning. If necessary, a local optimizer could be used

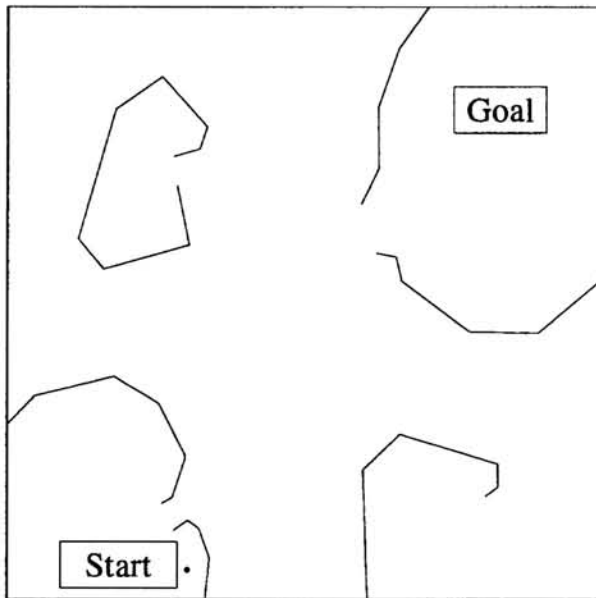

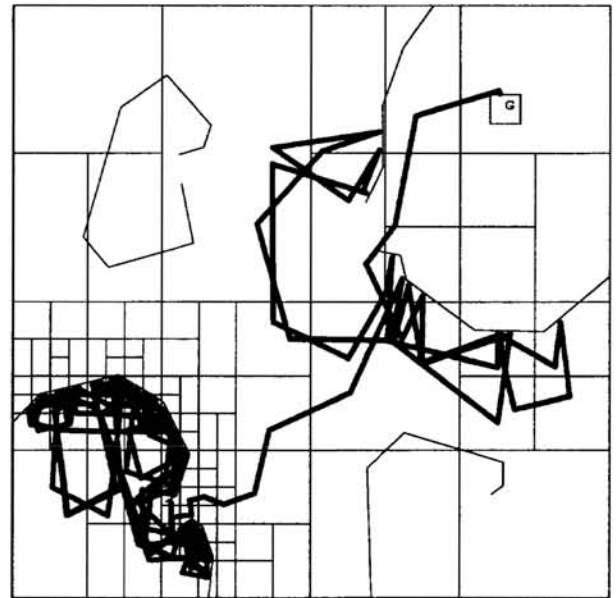

Figure 2: A 2-d maze problem. The point robot must find a path from start to goal without crossing any of the barrier lines. Remember that initially it does not know where any obstacles are, and must discover them by finding impassable states.

Figure 3: The path taken during the entire first trial. See text for explanation.

to refine this trajectory[2].

The system does not explore unnecessary areas. The barrier in the top left remains at low resolution because the system has had no need to visit there. Figures 5 and 6 show what happens when we now start the system inside this barrier.

Figure 7 shows a 3-d state space problem. If a standard grid were used, this would need an enormous number of states because the solution requires detailed three-point-turns. Parti-game's total exploration took 18 times as much movement as one run of the final path obtained.

Figure 8 shows a 4-d problem in which a ball rolls around a tray with steep edges. The goal is on the other side of a ridge. The maximum permissible force is low, and so greedy strategies, or globally linear control rules, get stuck in a limit cycle. Parti-game's solution runs to the other end of the tray, to build up enough velocity to make it over the ridge. The exploration-length versus final-path-length ratio is 24.

Figure 9 shows a 9-joint snake-like robot manipulator which must move to a specified configuration on the other side of a barrier. Again, no initial model is given: the controller must learn it as it explores. It takes seven trials before fixing on the solution shown. The exploration-length versus final-path-length ratio is 60.

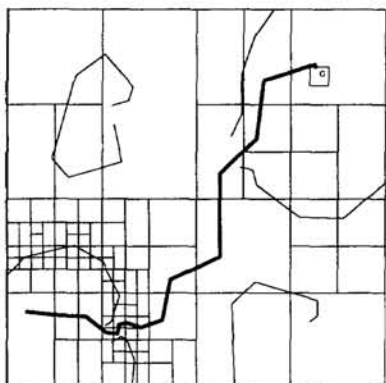

Figure 4: The second trial.

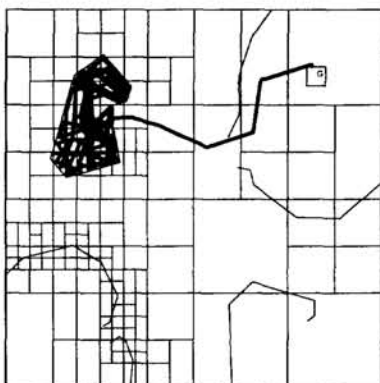

Figure 5: Starting inside the top left barrier.

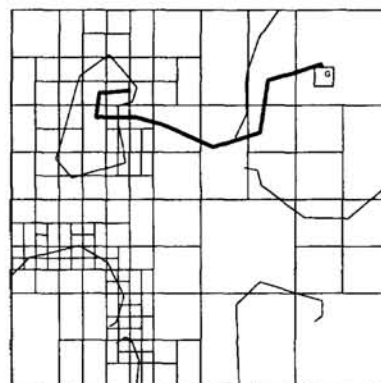

Figure 6: The trial after that.

Figure 7: A problem with a planar rod being guided past obstacles. The state space is three-dimensional: two values specify the position of the rod's center, and the third specifies the rod's angle from the horizontal. The angle is constrained so that the pole's dotted end must always be below the other end. The pole's center may be moved a short distance (up to 1/40 of the diagram width) and its angle may be altered by up to 5 degrees, provided it does not hit a barrier in the process. Parti-game converged to the path shown below after two trials. The partitioning lines on the solution diagram only show a 2-d slice of the full $k$d-tree.

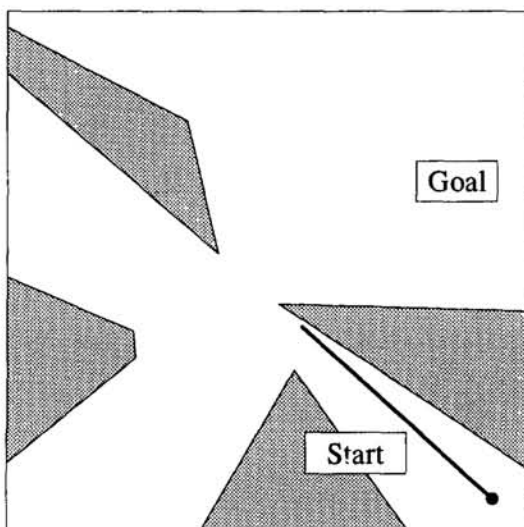

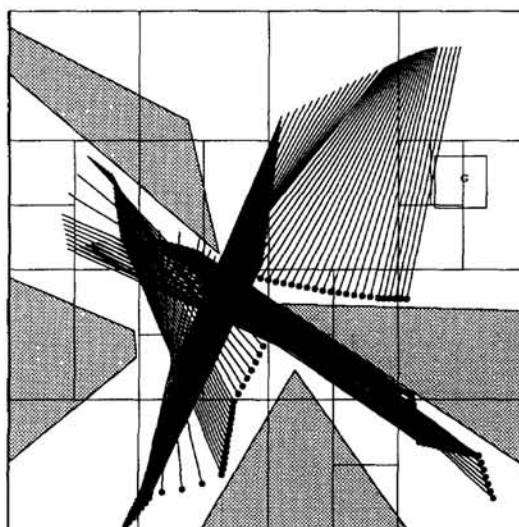

| Trials | 1 | 2 | 3 | 4 | 5 | 6 | 7 | 8 | 9 | 10 |
|---|---|---|---|---|---|---|---|---|---|---|
| Steps | 2975 | 189 | 187 | no further | | | | | | |
| Partitions | 149 | 149 | 149 | change | | | | | | |

Figure 8: A puck sliding over a hilly surface (hills shown by contours below: the surface is bowl shaped, with the lowest points nearest the center, rising steeply at the edges). The state space is four-dimensional: two position and two velocity variables. The controls consist of a force which may be applied in any direction, but with bounded magnitude. Convergence time was two trials.

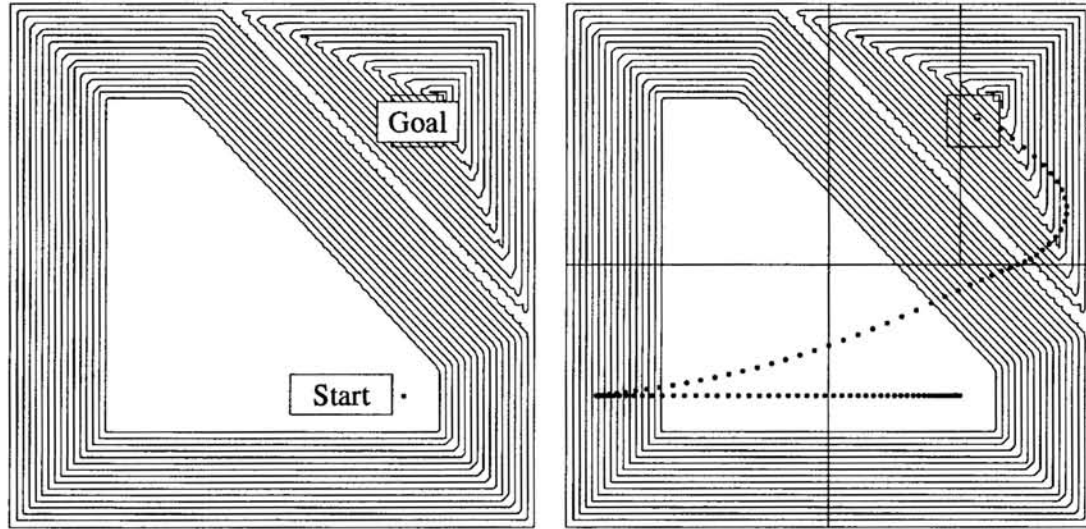

| Trials | 1 | 2 | 3 | 4 | 5 | 6 | 7 | 8 | 9 | 10 |
|---|---|---|---|---|---|---|---|---|---|---|
| Steps | 2609 | 115 | no further | | | | | | | |
| Partitions | 13 | 13 | change | | | | | | | |

Figure 9: A nine-degree-of-freedom planar robot must move from the shown start configuration to the goal. The solution entails curling, rotating and then uncurling. It may not intersect with any of the barriers, the edge of the workspace, or itself. Convergence occurred after seven trials.

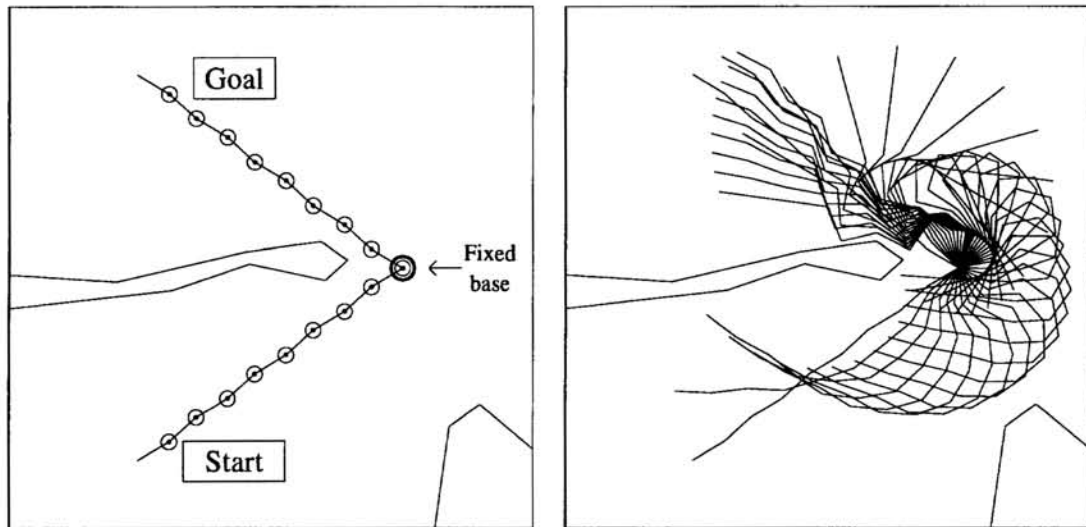

| Trials | 1 | 2 | 3 | 4 | 5 | 6 | 7 | 8 | 9 | 10 |
|---|---|---|---|---|---|---|---|---|---|---|
| Steps | 1090 | 430 | 353 | 330 | 739 | 200 | 52 | no further | | |
| Partitions | 41 | 66 | 67 | 69 | 78 | 85 | 85 | change | | |

# 5   DISCUSSION

Possible extensions include:

- Splitting criteria that lay down splits between trajectories with spatially distinct outcomes.
- Allowing humans to provide hints by permitting user-specified controllers ("behaviors") as extra high-level actions.
- Coalescing neighboring partitions that mutually agree.

We finish by noting a promising sign involving a series of snake robot experiments with different numbers of links (but fixed total length). Intuitively, the problem should get easier with more links, but the curse of dimensionality would mean that (in the absence of prior knowledge) it becomes exponentially harder. This is borne out by the observation that random exploration with the three-link arm will stumble on the goal eventually, whereas the nine link robot cannot be expected to do so in tractable time. However, Figure 10 indicates that as the dimensionality rises, the amount of exploration (and hence computation) used by parti-game does not rise exponentially. Real-world tasks may often have the same property as the snake example: the complexity of the ultimate task remains roughly constant as the number of degrees of freedom increases. If so, we may have uncovered the Achilles' heel of the curse of dimensionality.

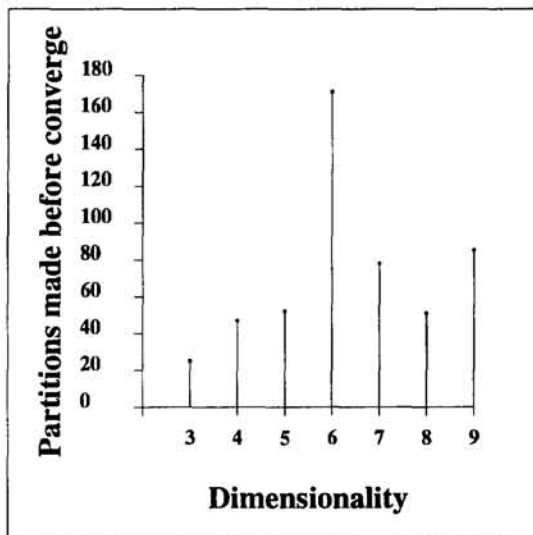

Figure 10: The number of partitions finally created against degrees of freedom for a set of snake-like robots. The $kd$-trees built were all highly non-uniform, typically having maximum depth nodes of twice the dimensionality. The relation between exploration time and dimensionality (not shown) had a similar shape.

## Footnotes

[1]More intelligent splitting criteria are under investigation.

[2]Another method is to increase the resolution along the trajectory [Moore, 1991].

# References

[Barto *et al.*, 1991] A. G. Barto, S. J. Bradtke, and S. P. Singh. Real-time Learning and Control using Asynchronous Dynamic Programming. Technical Report 91-57, University of Massachusetts at Amherst, August 1991.

[Bellman, 1957] R. E. Bellman. *Dynamic Programming*. Princeton University Press, Princeton, NJ, 1957.

[Chapman and Kaelbling, 1991] D. Chapman and L. P. Kaelbling. Learning from Delayed Reinforcement In a Complex Domain. Technical Report, Teleos Research, 1991.

[Dayan and Hinton, 1993] P. Dayan and G. E. Hinton. Feudal Reinforcement Learning. In S. J. Hanson, J. D Cowan, and C. L. Giles, editors, *Advances in Neural Information Processing Systems 5*. Morgan Kaufmann, 1993.

[Friedman *et al.*, 1977] J. H. Friedman, J. L. Bentley, and R. A. Finkel. An Algorithm for Finding Best Matches in Logarithmic Expected Time. *ACM Trans. on Mathematical Software*, 3(3):209–226, September 1977.

[Moore and Atkeson, 1993] A. W. Moore and C. G. Atkeson. Prioritized Sweeping: Reinforcement Learning with Less Data and Less Real Time. *Machine Learning*, 13, 1993.

[Moore, 1991] A. W. Moore. Variable Resolution Dynamic Programming: Efficiently Learning Action Maps in Multivariate Real-valued State-spaces. In L. Birnbaum and G. Collins, editors, *Machine Learning: Proceedings of the Eighth International Workshop*. Morgan Kaufman, June 1991.

[Samuel, 1959] A. L. Samuel. Some Studies in Machine Learning using the Game of Checkers. *IBM Journal on Research and Development*, 3, 1959. Reprinted in E. A. Feigenbaum and J. Feldman, editors, *Computers and Thought*, McGraw-Hill, 1963.

[Simons *et al.*, 1982] J. Simons, H. Van Brussel, J. De Schutter, and J. Verhaert. A Self-Learning Automaton with Variable Resolution for High Precision Assembly by Industrial Robots. *IEEE Trans. on Automatic Control*, 27(5):1109–1113, October 1982.

[Singh, 1993] S. Singh. Personal Communication.  , 1993.

[Sutton, 1984] R. S. Sutton. Temporal Credit Assignment in Reinforcement Learning. Phd. thesis, University of Massachusetts, Amherst, 1984.

[Watkins, 1989] C. J. C. H. Watkins. Learning from Delayed Rewards. PhD. Thesis, King's College, University of Cambridge, May 1989.